# Hippocampal Contributions to Control: The Third Way

**Máté Lengyel**
Collegium Budapest Institute for Advanced Study
2 Szentháromság u, Budapest, H-1014, Hungary
and
Computational & Biological Learning Lab
Cambridge University Engineering Department
Trumpington Street, Cambridge CB2 1PZ, UK
`lmate@gatsby.ucl.ac.uk`

**Peter Dayan**
Gatsby Computational Neuroscience Unit, UCL
17 Queen Square, London WC1N 3AR, UK
`dayan@gatsby.ucl.ac.uk`

## Abstract

Recent experimental studies have focused on the specialization of different neural structures for different types of instrumental behavior. Recent theoretical work has provided normative accounts for why there should be more than one control system, and how the output of different controllers can be integrated. Two particlar controllers have been identified, one associated with a forward model and the prefrontal cortex and a second associated with computationally simpler, habitual, actor-critic methods and part of the striatum. We argue here for the normative appropriateness of an additional, but so far marginalized control system, associated with episodic memory, and involving the hippocampus and medial temporal cortices. We analyze in depth a class of simple environments to show that episodic control should be useful in a range of cases characterized by complexity and inferential noise, and most particularly at the very early stages of learning, long before habitization has set in. We interpret data on the transfer of control from the hippocampus to the striatum in the light of this hypothesis.

## 1   Introduction

What use is an episodic memory? It might seem that the possibility of a fulminant recreation of a former experience plays a critical role in enabling us to act appropriately in the world [1]. However, why should it be better to act on the basis of the recollection of single happenings, rather than the seemingly normative use of accumulated statistics from multiple events? The task of building such a statistical model is normally the dominion of semantic memory [2], the other main form of declarative memory. Issues of this kind are frequently discussed under the rubric of multiple memory systems [3, 4]; here we consider it from a normative viewpoint in which memories are directly used for control.

Our answer to the initial question is the computational challenge of using a semantic memory as a forward model in sequential decision tasks in which many actions must be taken before a goal is reached [5]. Forward and backward search in the tree of actions and consequent states (*ie* model-based reinforcement learning [6]) in such domains impose crippling demands on working memory

(to store partial evaluations) and it may not even be possible to expand out the tree in reasonable times. If we think of the inevitable resulting errors in evaluation as a form of computational noise or uncertainty, then the use of the semantic memory for control will be expected to be subject to substantial error. The main task for this paper is to explore and understand the circumstances under which episodic control, although seemingly *less* efficient in its use of experience, should be expected to be *more* accurate, and therefore be evident both psychologically and neurally.

This argument about episodic control exactly parallels one recently made for habitual or cached control [5]. Model-free reinforcement learning methods, such as $\mathcal{Q}$-learning [7] are computationally trivial (and therefore accurate) at the time of use, since they learn state-value functions or state-action-value functions that cache the results of the expensive and difficult search. However, model-free methods learn through a form of bootstrapping, which is known to be inefficient in the use of experience. It is therefore optimal to employ cached control rather than model-based control only after *sufficient* experience, when the inaccuracy of the former over learning is outweighed by the computational noise induced in using the latter. The exact tradeoff depends on the prior statistics over the possible tasks.

We will show that in general, just as model-free control is better than model-based control after substantial experience, episodic control is better than model-based control after only very limited experience. For some classes of environments, these two other controllers significantly squeeze the domain of optimal use of semantic control.

This analysis is purely computational. However, it has psychological and neural implications and associations. It was argued [5] that the transition from model-based to model-free control explains a wealth of psychological observations about the transition over the course of learning from goal-directed control (which is considered to be model-based) to habitual control (which is model-free). In turn, this is associated with an apparent functional segregation between the dorsolateral prefrontal cortex and dorsomedial striatum, implementing model-based control, and the dorsolateral striatum (and its neuromodulatory inputs), implementing model-free control. Exactly how the uncertainties associated with these two types of control are calculated is not clear, although it is known that the prelimbic and infralimbic cortices are somehow involved in arbitration. The psychological construct for episodic control is obvious; its neural realization is likely to be the hippocampus and medial temporal cortical regions. How arbitration might work for this third controller is also not clear, although there have been suggestions on how uncertainty may be represented neurally in the hippocampus [8]. There is also evidence for the transfer of control from hippocampal to striatal structures over the course of learning [9, 10] suggesting that arbitration might happen, but unfortunately, in these studies, the possibility of an additional step via dorsolateral prefrontal cortex was not fully tested.

In this paper, we explore the nature and (f)utility of episodic control. Section 2 describes the simple tree-structured Markov decision problems that we use to illustrate and quantitate our arguments. Section 3 provides a detailed, albeit approximate, analysis of uncertainty and learning in these environments. Finally, section 4 uses these analytical methods and simulations to study the episodic/forward model tradeoff.

## 2  Paradigm for analysis

We seek to analyse computational and statistical trade-offs that arise in choosing actions that maximize long-term rewards in sequential decision making problems. The trade-offs originate in uncertainties associated with learning and inference. We characterize these tasks as Markov decision processes (MDPs) [6] whose transition and reward structure are initially unknown by the subject, but are drawn from a parameterized prior that is known.

The key question is how well different possible control strategies can perform given this prior and a measured amount of experience. Like [11], we simplify *exploration* using a form of parallel sampling model in order to focus on the ability of controllers to *exploit* knowledge extracted about an environment. Performance is naturally measured using the average reward that would be collected in a trial; this average is then itself averaged over draws of the MDP and the stochasticity associated with the exploratory actions. We analyse three controllers: a model-based controller without computational noise, which provides a theoretical upper limit on performance, a realistic model-based

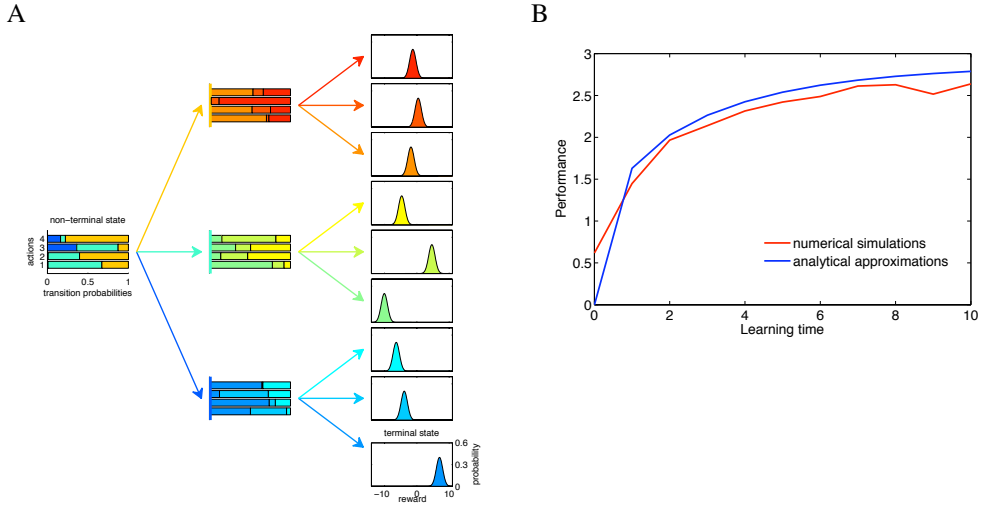

Figure 1: **A,** An example tree-structured MDP, with depth $D = 2$, branching factor $B = 3$, and $A = 4$ available actions in each non-terminal state. The horizontal stacked bars in the boxes of the left and middle column show the transition probabilities for different actions at non-terminal states, color coded by the successor states to which they lead (matching the color of the corresponding arrows). Transition probability distributions are iid. according to a Dirichlet distribution whose parameters are all $1$. Gaussians in the right column show the reward distributions at terminal states. Each has unit variance and a mean which is drawn iid. from a normal distribution of mean $\mu_{\bar{r}} = 0$ and standard deviation $\sigma_{\bar{r}} = 5$. All parameters in later figures are the same, unless otherwise noted. **B,** Validating the analytical approximations by numerical simulations ($A = 3$).

controller with computational noise that we regard as the model of semantic memory-based control, and an 'episodic controller'.

We concentrate on a simple subset of MDPs, namely 'tree-structured MDPs' (tMDPs), which are illustrated in Figure 1A (and defined formally in the Supporting Material). We expect the qualitative characteristics of our findings to apply for general MDPs; however, we used tMDPs since they represent a first-order, analytically tractable, approximation of the general problem presented by *any* MDP at a given decision point if it is unfolded in time (*ie* a decision tree with finite time-horizon). Actions lead to further states (and potentially rewards), from where further possible actions and thus states become available, and so on. The key difference is that in a general MDP, a state can be revisited several times even within the same episode, which is impossible in a tMDP. Thus, our approach neglects correlations between values of future states. This is formally correct in the limit of infinitely diluted MDPs, but is otherwise just an approximation.

## 3   The model-based controller

In our paradigm, the task for the model-based controllers is to use the data from the exploratory trials to work out posterior distributions over the unknown transition and reward structure of the tMDP, and then report the best action at each state. It is well known that actually doing this is radically intractable. However, to understand the tradeoffs between different controllers, we only need to analyze the expected return from doing so, averaging over all the random quantities. One of the technical contributions of this work is the set of analytically- and empirically-justified approximations to those averages (which are presented in the Supplementary Material), based on the assumed knowledge of the parameters governing the generation of the tMDP, and as a function of the amount of exploratory experience.

We proceed in three stages. First, we consider the model-based controller in the case that it has experienced so many samples that the parameters of the tMDP are known exactly. This provides an (approximate) upper bound on the expected performance of any controller. Second, we approximate

the impact of incomplete exploration by corrupting the controller by an aliquot of noise whose magnitude is determined by the parameters of the problem. Finally, we approximate the additionally deleterious effect of limited computational resources by adding an assumed induced bias and extra variance.

The first step is to calculate the asymptotic performance when infinitely many data have been collected. In this limit, transition probabilities and reward distributions can be treated as known quantities. Critical to our analysis is that the independence and symmetry properties of regular tMDPs imply that we mostly need only analyze a single 'sub-treelet' of the tree (one non-terminal state and its successor states), from which the results generalise to the whole tree by recursion. In the case of the asymptotic analysis, this recursive formulation turns out to allow for a closed-form solution for the mean $\mu$ and variance $\sigma^2$ of an approximate Gaussian distribution characterizing the average value of one full tree traversal starting from the root node:

$$\mu = \mu_{\bar{r}} + \frac{1 - \lambda_2^{D/2}}{1 - \lambda_2^{1/2}} \lambda_1 \sigma_{\bar{r}} \qquad \sigma 2 = \lambda_2^D \sigma_{\bar{r}}^2 \qquad (1)$$

where $\mu_{\bar{r}}$ and $\sigma_{\bar{r}}^2$ are the mean and variance of the normal distribution from which the means of the reward distributions at the terminal states are drawn, and $0 \leq \lambda_1, \lambda_2 \leq 1$ are constants that depend on the other parameters of the tMDP. This calculation depends on characterizing order statistics of multivariate Gaussian distributions which are *equicorrelated* (all the off-diagonal terms of the covariance matrix are the same) [12]. Equation 1 is actually an interesting result in and of itself – it indicates the extent to which the controller can take advantage of the variability $\mu - \mu_{\bar{r}} \propto \sigma_{\bar{r}}$ in boosting its expected return from the root node as a function of the depth of the tree.

The second step is to observe that we expect the benefits of episodic control to be most apparent given very limited exploratory experience. To make analytical progress, we are forced to make the significant assumption that the effects of this can be modeled by assuming that the controller does not have access to the true values of actions, but only to 'noisy' versions. This 'noise' comes from the fact that computing the values of different actions is based on *estimates* of transition probability and reward distributions. These estimates are inherently stochastic themselves, as they are based on stochastic experience. We have been able to show that the form of the resulting 'noise' in the action values can have the effect of scaling down the true values of actions at states by a factor $\phi_1$ and adding extra noise $\phi_2$. Although we were unable to find a closed-form solution for the effects of $\phi_1$ and $\phi_2$ on the performance of the controller, a recursive analytical formulation, though involved, is still possible (see Supporting Material).

Figure 1B shows the learning curve for the model-based controller computed using our analytical predictions (blue line) and using exhaustive numerical simulations (red line, average performance in 100 sample tMDPs, with the learning process rerun 100 times in each). The inaccuracies entailed by our approximations are tolerable (also for other parameters; simulations not shown), and so from this point we use those to analyse the performance of the optimal, model-based controller.

The dark blue solid curve in figure 2A (labelled $\eta_2 = 0$) shows the performance of model-based control as a function of the number of exploration samples (the equivalent of the dark blue curve in figure 1B, but for $A = 4$ rather than $A = 3$). For comparison, the dashed line shows the asymptotic expected value. The slight decrease in the approximate value arises because the approximations become slightly looser as the noise gets less; however, once again we have been able to show (simulations not shown) that our analysis is highly accurate compared with extensive actual samples.

The final step is to model the effects of the computational complexity of the model-based controller on performance arising from the severe demands it places on such facets as working memory. These necessitate pruning (*ie* ignoring parts of the decision tree), or sub-sampling, or some other such approximation. We treat the effects of all approximations by forcing the controller to have access to only noisy versions of the (exploration-limited) action values. Just as for incomplete exploration, we model the noise as a combination of downscaling the true action values by a parameter $\eta_1$ and adding excess variability $\eta_2$. Note that whereas the terms $\phi_1, \phi_2$ characterizing the effects of learning are determined by the number of samples; $\eta_1, \eta_2$ are set by hand to capture the assumed effects on inference of the computational complexity. The asymptotic values of the curves in figure 2A for various values of $\eta_2$ (for all of them, $\eta_1 = 1$) demonstrate the effects of inferential noise on performance.

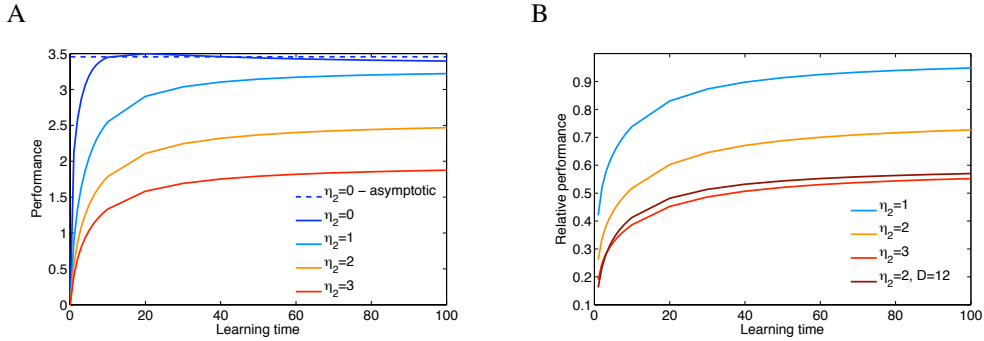

Figure 2:

**A,** Learning curves for the model-based controller at different levels of computational noise: $\eta_1 = 1$, $\eta_2$ is increased from 0 to 3. The approximations used for computing these curves are less accurate in the low-noise limit, hence the paradoxical slight decrease in the performance of the perfect controller (without noise) at the end of learning. The dashed line shows the asymptotic approximation which is more accurate in this limit, demonstrating that the inaccuracy of the experience-dependent approximation is not disastrous. **B,** Performance of noisy controllers normalized by that of the perfect controller in the same environment at the same amount of experience. The brown line corresponds to a more difficult environment with greater depth. Note that 'learning time' is measured by the number of times *every* state-action pair has been sampled. Thus decreased performance shown in the more complex environment is *not* due to the increased sparsity of experience.

So far, we have separately considered the effects of computational noise and uncertainty due to limited experience. In reality, both factors affect the model-based controller. The full plots in figure 2A, B show the interaction of these two factors (figure 2B shows the same data as figure 2A, but scaled to the performance of the noise-free controller for the given amount of experience). Computational noise not only makes the asymptotic performance worse, by simply down-scaling average rewards, but it also makes learning *effectively slower*. This is because the adverse effects of computational noise depend on the differences between the values of possible actions. If these values appear to be widely different, then computational noise will still preserve their order, and thus the one that is truly best is still likely to be chosen. However, if action values appear roughly the same, then a little noise can easily change their ordering and make the controller choose a suboptimal one. Little experience only licenses small apparent differences between values, and this boosts the corrupting effect of the inferential noise. Given more experience, the controller increasingly learns to make distinctions between different actions that looked the same *a priori*.

Thus, while earlier work suggested that model-based control will be superior at the limit of few exploratory samples due to the unsurpassable data-efficiency of optimal statistical inference [5], we show here that in the really low data limit another factor cripples its performance: the amplified influence of computational noise. How much experience constitutes 'little' and how much noise counts as 'much' is of course relative to the complexity of the environment.

## 4 Episodic control

If model-based control is indeed crippled by computational noise given limited exploration, could there be an effective alternative? Although outside the scope of our formal analysis, this is particularly important given the ubiquity of non-stationary environments [13], for which the effects of continual change bound the effective number of exploratory samples. That the cache-based or habitual controller is even *worse* in this limit (since it learns by bootstrapping) was a main rationale for the uncertainty-based account of the transfer from goal-directed to habitual control suggested by Daw *et al* [5]. Thus the habitual controller cannot step into the breach.

It is here that we expect episodic control to be most useful. Intuitively, if a subject has experienced a complex environment just a few times, and found a sequence of actions that works reasonably well, then, provided that exploitation is at a premium over exploration, it seems obvious for the subject

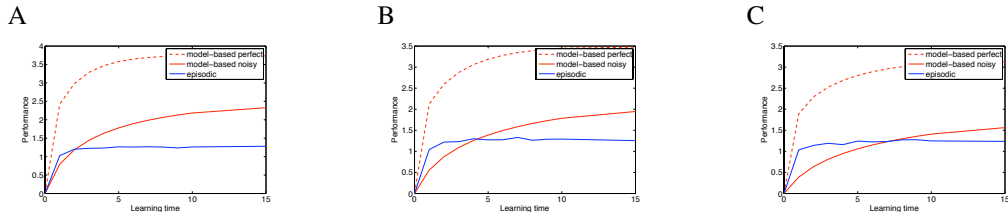

Figure 3: Episodic vs. model-based control. Solid red line shows the performance of noisy model-based control ($\eta_2 = 2$), blue line shows that of episodic control. Dashed red line shows the case of perfect model-based control which constitutes the best performance that could possibly be achieved. The branching factor of the environment increased from $B = 2$ (**A**), $B = 3$ (**B**) to $B = 4$ (**C**).

just to repeat exactly those actions, rather than trying to build and use a complex model. This act of replaying a particular sequence of events from the past is exactly an instance of episodic control.

More specifically, we employ an extremely simple model of episodic memory, and assume that each time the subject experiences a reward that is considered large enough (larger than expected *a priori*) it stores the specific sequence of state-action pairs leading up to this reward, and tries to follow such a sequence whenever it stumbles upon a state included in it. If multiple successful sequences are available for the same state, the one that yielded maximal reward is followed. We expect such a strategy to be useful in the low data limit because, unlike in cache-based control, there is no issue of bootstrapping and temporal credit assignment, and unlike in model-based control, there is no exhaustive tree-search involved in action selection. Of course its advantages will be ultimately counteracted by the haphazardness of using single samples that are 'adequate', but by that time the other controllers can take over.

Although we expect our approximate analytical methods to provide some insight into its characteristics, we have so far only been able to use simulations to study the episodic controller in the usual class of tMDPs. Comparing the blue (episodic) and red (model-based, but noisy; $\eta_2 = 2$) curves, in figure 3A-C, it is apparent that episodic control indeed outperforms noisy model-based control in the low data limit. The dashed curves show the performance of the idealized model-based controller that is noise-free. This emphasizes the arbitrariness of our choice of noise level – the greater the noise, the longer the dominance of episodic control. However, in complicated environments, even very small amounts of noise are near catastrophic for model-based control (see brown line in Fig. 2B), and so this issue is not nugatory.

The progression of the learning curves in figure 3A-C make the same point a different way. They show what happens as the complexity of the environment is increased by increasing the branching factor. At the same level of computational noise, episodic control supplants model-based control for increasing volumes of exploratory samples. We expect that the same is true if the complexity of the environment is increased by increasing the depth of the tree ($D$) instead, or as well.

Figure 3A-C also makes the point that the asymptotic performance of the episodic controller is rather poor, and is barely improved by extra learning. A smarter episodic strategy, perhaps involving reconsolidation to eliminate unfortunate sample trajectories, might perform more competently.

## 5   Discussion

An episodic controller operates by remembering for each state the single action that led to the best outcome so far observed. Here, we studied the nature and benefits of episodic control. This controller is statistically inefficient for solving Markov decision problems compared with the normative strategy of building a statistical forward model of the transitions and outcomes, and searching for the optimal action. However, episodic control is computationally very straightforward, and therefore does not suffer from any excess uncertainty or noise arising from the severe calculational and search complexities of the forward model. This implies that it can best forward model control under various circumstances.

To explore this, we first characterized a class of regular tree-structured Markov decision problems using four critical parameters – the depth of the tree; the fan-out from each state; the number of actions per state, and the characteristic (Dirichlet) statistics of the transitions consequent on each action. We then used theoretical and empirical methods to analyze the statistical structure of control based on a forward model in the face of limited data. We showed that this control can readily be outperformed by an episodic controller which does not suffer from computational inaccuracy, at least in the particular limits of high task complexity and significant inferential noise in the model-based controller. We also showed how the noise in the latter has a particularly pernicious effect on the course of learning, corrupting the choice between actions whose values appear, because of limited experience, closer than they actually are.

Our results are obviously partial. In particular, the constraint of using a regular tree-structured MDP is much too severe – given the intuition from the results above, we can now consider more conventional MDPs that better model the classes of experiments that have probed the transfer of control. Further, it would be important to consider models of exploration more general than the parallel sampler, which provides homogeneous sampling of state-action pairs. The particular challenge is when exploration and exploitation are coupled, as then all the samples become interdependent in a gordian manner.

Our analysis paralleled that of [5], who showed that the noisy forward-model controller is also beaten by a cached (actor-critic-like) controller in the opposite limit of substantial experience in an environment. The cached controller is also computationally straightforward, but relies on a completely different structure of learning and inference.

In psychological terms, the episodic controller is best thought of as being goal-directed, since the ultimate outcome forms part of the episode that is recalled. Unfortunately, this makes it difficult to distinguish behaviorally from goal-directed control resulting from the forward model. In neural terms, the episodic controller is likely to rely on the very well investigated systems involved in episodic memory, namely the hippocampus and medial temporal cortices. Importantly, there is direct evidence of the transfer of control from hippocampal to striatal structures over the course of learning [9, 10], and there is some evidence that episodic and habitual control can be simultaneously active. Unfortunately, there are few data [14] on structures that might control the competition or transfer process, and no test as to whether there is an intermediate phase in which prefrontal mechanisms instantiating the forward model might be dominant. Predictions from our work associated with this are the most ripe for experimental test.

This paper is an extended answer to the question of the computational benefit of episodic memory, which, crudely speaking, stores particular samples, over semantic memory, which stores probability distributions. It is, of course, not the only answer – for instance, subjects that cache are obviously better off remembering exactly where in particular they stored food [15] than having to search all the places that are likely under a (semantic) prior. Equally, in game theoretic interactions between competitors, Nash equilibria are typically stochastic, and therefore seemingly excellent candidates for control based on a semantic memory. However, taking advantage of the flaws in an opponent require exactly remembering how its actions deviate from stationary statistics, for which an episodic memory is a most useful tool [16].

One potential caveat to our results is that methods associated with memory-based reasoning [17] (such as kernel density estimation) create a semantic memory from an episodic one, for instance by recalling all episodes close to a cue, and weighting them by a statistically-appropriate measure of their distance. This form of semantic memory can be seen as arising without any consolidation process whatsoever. However, although this method has its computational attractions, it is psychologically implausible since phenomena such as priming make it extremely difficult to recall multiple closely related samples from an episodic memory, let alone to do so in a statistically unbiased way (but see [18]).

In sum, we have provided a normative justification from the perspective of appropriate control for the episodic component of a multiple memory system. Pressing from a theoretical perspective is a richer understanding of the integration beyond mere competition, of the information residing in, and the decisions made by, all the systems involved in choice.

**Acknowledgements**

We are very grateful to Nathaniel Daw and Quentin Huys for helpful discussions. Funding was from the Gatsby Charitable Foundation (ML and PD), and the EU Framework 6 (IST-FET 1940) (ML).

**References**

[1] Dudai, Y. & Carruthers, M. The Janus face of Mnemosyne. *Nature* **434**, 567 (2005).

[2] Káli, S. & Dayan, P. Off-line replay maintains declarative memories in a model of hippocampal-neocortical interactions. *Nat. Neurosci.* **7**, 286–294 (2004).

[3] McClelland, J.L., McNaughton, B.L. & O'Reilly, R.C. Why there are complementary learning systems in the hippocampus and neocortex: insights from the successes and failures of connectionist models of learning and memory. *Psychol. Rev.* **102**, 419–457 (1995).

[4] White, N.M. & McDonald, R.J. Multiple parallel memory systems in the brain of the rat. *Neurobiol Learn Mem* **77**, 125–184 (2002).

[5] Daw, N.D., Niv, Y. & Dayan, P. Uncertainty-based competition between prefrontal and dorsolateral striatal systems for behavioral control. *Nat. Neurosci.* **8**, 1704–1711 (2005).

[6] Sutton, R.S. & Barto, A.G. *Reinforcement Learning* (MIT Press, 1998).

[7] Watkins, C.J.C.H. *Learning from Delayed Rewards*. PhD thesis, Cambridge University, (1989).

[8] Lengyel, M. & Dayan, P. Uncertainty, phase and oscillatory hippocampal recall. in *Advances in Neural Information Processing Systems 19* (eds. Schölkopf, B., Platt, J. & Hoffman, T.) 833–840 (MIT Press, Cambridge, MA, 2007).

[9] Packard, M.G. & McGaugh, J.L. Double dissociation of fornix and caudate nucleus lesions on acquisition of two water maze tasks: further evidence for multiple memory systems. *Behav. Neurosci.* **106**, 439–446 (1992).

[10] Poldrack, R.A. *et al.* Interactive memory systems in the human brain. *Nature* **414**, 546–550 (2001).

[11] Kearns, M. & Singh, S. Finite-sample convergence rates for Q-learning and indirect algorithms. in *Advances in Neural Information Processing Systems* Vol. 11 (eds. Kearns, M.S., Solla, S.A. & Cohn, D.A.), Vol. 11, 996–1002 (MIT Press, Cambridge, MA, 1999).

[12] Owen, D.B. & Steck, G.P. Moments of order statistics from the equicorrelated multivariate normal distribution. *Ann Math Stat* **33**, 1286–1291 (1962).

[13] Kording, K.P., Tenenbaum, J.B. & Shadmehr, R. The dynamics of memory as a consequence of optimal adaptation to a changing body. *Nat. Neurosci.* **10**, 779–786 (2007).

[14] Poldrack, R.A. & Rodriguez, P. How do memory systems interact? Evidence from human classification learning. *Neurobiol Learn Mem* **82**, 324–332 (2004).

[15] Clayton, N.S. & Dickinson, A. Episodic-like memory during cache recovery by scrub jays. *Nature* **395**, 272–274 (1998).

[16] Clayton, N.S., Dally, J.M. & Emery, N.J. Social cognition by food-caching corvids. the western scrub-jay as a natural psychologist. *Philos. Trans. R. Soc. Lond. B Biol. Sci.* **362**, 507–522 (2007).

[17] Stanfill, C. & Waltz, D. Toward memory-based reasoning. *Communications of the ACM* **29**, 1213–1228 (1986).

[18] Hintzman, D.L. MINERVA 2: A simulation model of human memory. *Behav Res Methods Instrum Comput* **16**, 96–101 (1984).

